# Multi-label Prediction via Sparse Infinite CCA

**Piyush Rai and Hal Daumé III**
School of Computing, University of Utah
{piyush,hal}@cs.utah.edu

## Abstract

Canonical Correlation Analysis (CCA) is a useful technique for modeling dependencies between two (or more) sets of variables. Building upon the recently suggested probabilistic interpretation of CCA, we propose a nonparametric, fully Bayesian framework that can automatically select the number of correlation components, and effectively capture the sparsity underlying the projections. In addition, given (partially) labeled data, our algorithm can also be used as a (semi)supervised dimensionality reduction technique, and can be applied to learn useful predictive features in the context of learning a set of related tasks. Experimental results demonstrate the efficacy of the proposed approach for both CCA as a stand-alone problem, and when applied to multi-label prediction.

## 1 Introduction

Learning with examples having multiple labels is an important problem in machine learning and data mining. Such problems are encountered in a variety of application domains. For example, in text classification, a document (e.g., a newswire story) can be associated with multiple categories. Likewise, in bio-informatics, a gene or protein usually performs several functions. All these settings suggest a common underlying problem: predicting multivariate responses. When the responses come from a discrete set, the problem is termed as multi-label classification. The aforementioned setting is a special case of multitask learning [6] when predicting each label is a task and all the tasks share a common source of input. An important characteristics of these problems is that the labels are not independent of each other but actually often have significant correlations with each other. A naïve approach to learn in such settings is to train a separate classifier for each label. However, such an approach ignores the label correlations and leads to sub-optimal performance [20].

In this paper, we show how Canonical Correlation Analysis (CCA) [11] can be used to exploit label relatedness, learning multiple prediction problems simultaneously. CCA is a useful technique for modeling dependencies between two (or more) sets of variables. One important application of CCA is in *supervised* dimensionality reduction, albeit in the more general setting where each example has several labels. In this setting, CCA on input-output pair $(\mathbf{X}, \mathbf{Y})$ can be used to project inputs $\mathbf{X}$ to a low-dimensional space directed by label information $\mathbf{Y}$. This makes CCA an ideal candidate for extracting useful predictive features from data in the context of multi-label prediction problems.

The classical CCA formulation, however, has certain inherent limitations. It is non-probabilistic which means that it cannot deal with missing data, and precludes a Bayesian treatment which can be important if the dataset size is small. An even more crucial issue is choosing the number of correlation components, which is traditionally dealt with by using cross-validation, or model-selection [21]. Another issue is the potential sparsity [18] of the underlying projections that is ignored by the standard CCA formulation.

Building upon the recently suggested probabilistic interpretation of CCA [3], we propose a nonparametric, fully Bayesian framework that can deal with each of these issues. In particular, the proposed model can automatically select the number of correlation components, and effectively capture the

sparsity underlying the projections. Our framework is based on the Indian Buffet Process [9], a nonparametric Bayesian model to discover latent feature representation of a set of observations. In addition, our probabilistic model allows dealing with missing data and, in the supervised dimensionality reduction case, can incorporate *additional* unlabeled data one may have access to, making our CCA algorithm work in a semi-supervised setting. Thus, apart from being a general, nonparametric, fully Bayesian solution to the CCA problem, our framework can be readily applied for learning useful predictive features from labeled (or *partially* labeled) data in the context of learning a set of related tasks.

This paper is organized as follows. Section 2 introduces the CCA problem and its recently proposed probabilistic interpretation. In section 3, we describe our general framework for *infinite* CCA. Section 4 gives a concrete example of an application (multi-label learning) where the proposed approach can be applied. In particular, we describe a fully supervised setting (when the test data is not available at the time of training), and a semi-supervised setting with partial labels (when we have access to test data at the time of training). We describe our experiments in section 5, and discuss related work in section 6 drawing connections of the proposed method with previously proposed ones for this problem. .

## 2  Canonical Correlation Analysis

Canonical correlation analysis (CCA) is a useful technique for modeling the relationships among a set of variables. CCA computes a low-dimensional *shared* embedding of a set of variables such that the correlations among the variables is maximized in the embedded space.

More formally, given a pair of variables $\mathbf{x} \in \mathbb{R}^{D_1}$ and $\mathbf{y} \in \mathbb{R}^{D_2}$, CCA seeks to find linear projections $\mathbf{u}_x$ and $\mathbf{u}_y$ such that the variables are maximally correlated in the projected space. The correlation coefficient between the two variables in the embedded space is given by

$$\rho = \frac{\mathbf{u}_x^T \mathbf{x}\mathbf{y}^T \mathbf{u}_y}{\sqrt{(\mathbf{u}_x^T \mathbf{x}\mathbf{x}^T \mathbf{u}_x)(\mathbf{u}_y^T \mathbf{y}\mathbf{y}^T \mathbf{u}_y)}}$$

Since the correlation is not affected by rescaling of the projections $\mathbf{u}_x$ and $\mathbf{u}_y$, CCA is posed as a constrained optimization problem.

$$\max_{\mathbf{u}_x, \mathbf{u}_y} \mathbf{u}_x^T \mathbf{x}\mathbf{y}^T \mathbf{u}_y, subject\ to : \mathbf{u}_x^T \mathbf{x}\mathbf{x}^T \mathbf{u}_x = 1, \mathbf{u}_y^T \mathbf{y}\mathbf{y}^T \mathbf{u}_y = 1$$

It can be shown that the above formulation is equivalent to solving the following generalized eigenvalue problem:

$$\begin{pmatrix} 0 & \mathbf{\Sigma_{xy}} \\ \mathbf{\Sigma_{yx}} & 0 \end{pmatrix} \begin{pmatrix} \mathbf{u_x} \\ \mathbf{u_y} \end{pmatrix} = \rho \begin{pmatrix} \mathbf{\Sigma_{xx}} & 0 \\ 0 & \mathbf{\Sigma_{yy}} \end{pmatrix} \begin{pmatrix} \mathbf{u_x} \\ \mathbf{u_y} \end{pmatrix}$$

where $\mathbf{\Sigma}$ denotes the covariance matrix of size $D \times D$ (where $D = D_1 + D_2$) obtained from the data samples $\mathbf{X} = [\mathbf{x}_1, \ldots, \mathbf{x}_n]$ and $\mathbf{Y} = [\mathbf{y}_1, \ldots, \mathbf{y}_n]$.

### 2.1  Probabilistic CCA

Bach and Jordan [3] gave a probabilistic interpretation of CCA by posing it as a latent variable model. To see this, let $\mathbf{x}$ and $\mathbf{y}$ be two random vectors of size $D_1$ and $D_2$. Let us now consider the following latent variable model

$$\begin{aligned} \mathbf{z} &\sim \mathcal{N}or(0, \boldsymbol{I}_K), \quad \min\{D_1, D_2\} \geq K \\ \mathbf{x} &\sim \mathcal{N}or(\mu_x + \mathbf{W}_x \mathbf{z}, \boldsymbol{\Psi}_x), \quad \mathbf{W}_x \in \mathbb{R}^{D_1 \times K}, \boldsymbol{\Psi}_x \succeq 0 \\ \mathbf{y} &\sim \mathcal{N}or(\mu_y + \mathbf{W}_y \mathbf{z}, \boldsymbol{\Psi}_y), \quad \mathbf{W}_y \in \mathbb{R}^{D_2 \times K}, \boldsymbol{\Psi}_y \succeq 0 \end{aligned}$$

Equivalently, we can also write the above as

$$[\mathbf{x}; \mathbf{y}] \sim \mathcal{N}or(\boldsymbol{\mu} + \mathbf{W}\mathbf{z}, \boldsymbol{\Psi})$$

where $\boldsymbol{\mu} = [\mu_x; \mu_y]$, $\mathbf{W} = [\mathbf{W}_x; \mathbf{W}_y]$, and $\boldsymbol{\Psi}$ is a block-diagonal matrix consisting of $\boldsymbol{\Psi}_x$ and $\boldsymbol{\Psi}_x$ on its diagonals. $[.;.]$ denotes row-wise concatenation. The latent variable $\mathbf{z}$ is shared between $\mathbf{x}$ and $\mathbf{y}$.

Bach and Jordan [3] showed that, given the maximum likelihood solution for the model parameters, the expectations $\mathbb{E}(\mathbf{z}|\mathbf{x})$ and $\mathbb{E}(\mathbf{z}|\mathbf{y})$ of the latent variable $\mathbf{z}$ lie in the same subspace that classical CCA finds, thereby establishing the equivalence between the above probabilistic model and CCA.

The probabilistic interpretation opens doors to several extension of the basic setup proposed in [3] which suggested a maximum likelihood approach for parameter estimation. However, it still assumes an *apriori* fixed number of canonical correlation components. In addition, another important issue is the sparsity of the underlying projection matrix which is usually ignored.

## 3 The Infinite Canonical Correlation Analysis Model

Recall that the CCA problem can be defined as $[\mathbf{x}; \mathbf{y}] \sim \mathcal{N}or(\mathbf{Wz}, \boldsymbol{\Psi})$ (assuming centered data). A crucial issue in the CCA model is choosing the number of canonical correlation components which is set to a fixed value in classical CCA (and even in the probabilistic extensions of CCA). In the Bayesian formulation of CCA, one can use the Automatic Relevance Determination (ARD) prior [5] on the projection matrix $\mathbf{W}$ that gives a way to select this number. However, it would be more appropriate to have a principled way to automatically figure out this number based on the data.

We propose a nonparametric Bayesian model that selects the number of canonical correlation components automatically. More specifically, we use the Indian Buffet Process [9] (IBP) as a nonparametric prior on the projection matrix $\mathbf{W}$. The IBP prior allows $\mathbf{W}$ to have an unbounded number of columns which gives a way to automatically determine the dimensionality $K$ of the latent space associated with $\mathbf{Z}$.

### 3.1 The Indian Buffet Process

The Indian Buffet Process [9] defines a distribution over infinite binary matrices, originally motivated by the need to model the latent feature structure of a given set of observations. The IBP has been a model of choice in variety of non-parametric Bayesian approaches, such as for factorial structure learning, learning causal structures, modeling dyadic data, modeling overlapping clusters, and several others [9].

In the latent feature model, each observation can be thought of as being explained by a set of latent features. Given an $N \times D$ matrix $\mathbf{X}$ of $N$ observations having $D$ features each, we can consider a decomposition of the form $\mathbf{X} = \mathbf{ZA} + \mathbf{E}$ where $\mathbf{Z}$ is an $N \times K$ binary feature-assignment matrix describing which features are present in each observation. $Z_{n,k}$ is 1 if feature $k$ is present in observation $n$, and is otherwise 0. $\mathbf{A}$ is a $K \times D$ matrix of feature scores, and the matrix $\mathbf{E}$ consists of observation specific noise. A crucial issue in such models is the choosing the number $K$ of latent features. The standard formulation of IBP lets us define a prior over the binary matrix $\mathbf{Z}$ such that it can have an unbounded number of columns and thus can be a suitable prior in problems dealing with such structures.

The IBP derivation starts by defining a finite model for $K$ many columns of a $N \times K$ binary matrix.

$$P(\mathbf{Z}) = \prod_{k=1}^{K} \frac{\frac{\alpha}{K}\Gamma(m_k + \frac{\alpha}{K})\Gamma(P - m_k - 1)}{\Gamma(P + 1 + \frac{\alpha}{K})} \qquad (1)$$

Here $m_k = \sum_i Z_{ik}$. In the limiting case, as $K \rightarrow \infty$, it as was shown in [9] that the binary matrix $\mathbf{Z}$ generated by IBP is equivalent to one produced by a sequential generative process. This equivalence can be best understood by a culinary analogy of customers coming to an Indian restaurant and selecting dishes from an infinite array of dishes. In this analogy, customers represent observations and dishes represent latent features. Customer 1 selects $Poisson(\alpha)$ dishes to begin with. Thereafter, each incoming customer $n$ selects an existing dish $k$ with a probability $m_k/N$, where $m_k$ denotes how many previous customers chose that particular dish. The customer $n$ then goes on further to additionally select $Poisson(\alpha/N)$ new dishes. This process generates a binary matrix $\mathbf{Z}$ with rows representing customer and columns representing dishes. Many real world datasets have a sparseness

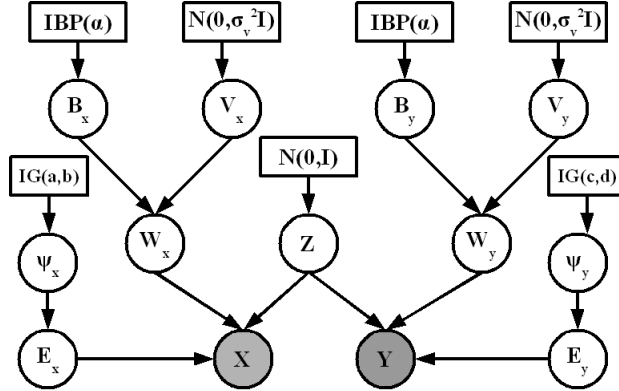

Figure 1: The graphical model depicts the fully supervised case when all variables X and Y are observed. The semisupervised case can have X and/or Y consisting of missing values as well. The graphical model structure remains the same

property which means that each observation depends only on a subset of all the $K$ latent features. This means that the binary matrix $\mathbf{Z}$ is expected to be reasonably sparse for many datasets. This makes IBP a suitable choice for also capturing the underlying sparsity in addition to automatically discovering the number of latent features.

## 3.2 The Infinite CCA Model

In our proposed framework, the matrix $\mathbf{W}$ consisting of canonical correlation vectors is modeled using an IBP prior. However since $\mathbf{W}$ can be real-valued and the IBP prior is defined only for binary matrices, we represent the $(D1 + D2) \times K$ matrix $\mathbf{W}$ as $(\mathbf{B} \odot \mathbf{V})$, where $\mathbf{B} = [\mathbf{B}_x; \mathbf{B}_y]$ is a $(D_1 + D_2) \times K$ binary matrix, $\mathbf{V} = [\mathbf{V}_x; \mathbf{V}_y]$ is a $(D_1 + D_2) \times K$ real-valued matrix, and $\odot$ denotes their element-wise (Hadamard) product. We place an IBP prior on $\mathbf{B}$ that automatically determines $K$, and a Gaussian prior on $\mathbf{V}$. Note that $\mathbf{B}$ and $\mathbf{V}$ have the same number of columns. Under this model, two random vectors $\mathbf{x}$ and $\mathbf{y}$ can be modeled as $\mathbf{x} = (\mathbf{B}_x \odot \mathbf{V}_x)\mathbf{z} + \mathbf{E}_x$ and $\mathbf{y} = (\mathbf{B}_y \odot \mathbf{V}_y)\mathbf{z} + \mathbf{E}_y$. Here $\mathbf{z}$ is shared between $\mathbf{x}$ and $\mathbf{y}$, and $\mathbf{E}_x$ and $\mathbf{E}_y$ are observation specific noise.

In the full model, $\mathbf{X} = [\mathbf{x}_1, \ldots, \mathbf{x}_N]$ is $D_1 \times N$ matrix consisting of $N$ samples of $D_1$ dimensions each, and $\mathbf{Y} = [\mathbf{y}_1, \ldots, \mathbf{y}_N]$ is another matrix consisting of $N$ samples of $D_2$ dimensions each. Here is the generative story for our basic model:

$$\begin{aligned}
\mathbf{B} &\sim \mathcal{IBP}(\alpha) \\
\mathbf{V} &\sim \mathcal{N}or(0, \sigma_v^2 \mathbf{I}), \quad \sigma_v \sim IG(a, b) \\
\mathbf{Z} &\sim \mathcal{N}or(0, I) \\
[\mathbf{X}; \mathbf{Y}] &\sim \mathcal{N}or(\mathbf{B} \odot \mathbf{V})\mathbf{Z}, \boldsymbol{\Psi}),
\end{aligned}$$

where $\boldsymbol{\Psi}$ is a diagonal matrix of size $D \times D$ where $D = (D_1 + D_2)$, with each diagonal entry having an inverse-Gamma prior..

Since our model is probabilistic, it can also deal the problem when $\mathbf{X}$ or $\mathbf{Y}$ have missing entries. This is particularly important in the case of supervised dimensionality reduction (i.e., $\mathbf{X}$ consisting of inputs and $\mathbf{Y}$ associated responses) when the labels for some of the inputs are unknown, making it a model for *semi-supervised* dimensionality reduction with partially labeled data. In addition, placing the IBP prior on the projection matrix $\mathbf{W}$ (via the binary matrix $\mathbf{B}$) also helps in capturing the sparsity in $\mathbf{W}$ (see results section for evidence).

## 3.3 Inference

We take a fully Bayesian approach by treating everything at latent variables and computing the posterior distributions over them. We use Gibbs sampling with a few Metropolis-Hastings steps to do inference in this model.

In what follows, $\mathbf{D}$ denotes the data $[\mathbf{X}; \mathbf{Y}]$, $\mathbf{B} = [\mathbf{B}_x; \mathbf{B}_y]$, and $\mathbf{V} = [\mathbf{V}_x; \mathbf{V}_y]$

**Sampling B:** Sampling the binary IBP matrix $\mathbf{B}$ consists of sampling existing dishes, proposing new dishes and accepting or rejecting them based on the acceptance ratio in the associated M-H step. For sampling existing dishes, an entry in $\mathbf{B}$ is set as 1 according to $p(B_{ik} = 1|\mathbf{D}, B_{-ik}, \mathbf{V}, \mathbf{Z}, \boldsymbol{\Psi}) \propto \frac{m_{-i,k}}{D} p(\mathbf{D}|\mathbf{B}, \mathbf{V}, \mathbf{F}, \boldsymbol{\Psi})$ whereas it is set as 0 according to $p(B_{ik} = 0|\mathbf{D}, B_{-ik}, \mathbf{V}, \mathbf{Z}, \boldsymbol{\Psi}) \propto \frac{D - m_{-i,k}}{D} p(\mathbf{D}|\mathbf{B}, \mathbf{V}, \mathbf{Z}, \boldsymbol{\Psi})$. $m_{-i,k} = \sum_{j \neq i} B_{jk}$ is how many other customers chose dish $k$.

For sampling new dishes, we use an M-H step where we simultaneously propose $\eta = (K^{new}, V^{new}, Z^{new})$ where $K^{new} \sim Poisson(\alpha/D)$. We accept the proposal with an acceptance probability given by $a = \min\{1, \frac{p(rest|\eta^*)}{p(rest|\eta)}\}$. Here, $p(rest|\eta)$ is the probability of the data given parameters $\eta$. We propose $V^{new}$ from its prior (Gaussian) but, for faster mixing, we propose $Z^{new}$ from its posterior.

**Sampling V:** We sample the real-valued matrix $\mathbf{V}$ from its posterior $p(V_{i,k}|\mathbf{D}, \mathbf{B}, \mathbf{Z}, \boldsymbol{\Psi}) \propto \mathcal{N}or(V_{i,k}|\mu_{i,k}, \Sigma_{i,k})$, where $\Sigma_{i,k} = (\sum_{n=1}^N \frac{Z_{k,n}^2}{\Psi_i} + \frac{1}{\sigma_v^2})^{-1}$ and $\mu_{i,k} = \Sigma_{i,k}(\sum_{n=1}^N A_{k,n} D_{i,k}^*) \Psi_i^{-1}$. We define $D_{i,k}^* = D_{i,n} - \sum_{l=1, l \neq k}^K (B_{i,l} V_{i,l}) Z_{l,n}$. The hyperparameter $\sigma_v$ on $\mathbf{V}$ has an inverse-gamma prior and posterior also has the same form. Note that the number of columns in $\mathbf{V}$ is the same as number of columns in the IBP matrix $\mathbf{B}$.

**Sampling Z:** We sample for $\mathbf{Z}$ from its posterior $p(\mathbf{Z}|\mathbf{D}, \mathbf{B}, \mathbf{V}, \boldsymbol{\Psi}) \propto \mathcal{N}or(\mathbf{Z}|\mu, \Sigma)$ where $\mu = \mathbf{W}^{\mathbf{T}}(\mathbf{W}\mathbf{W}^{\mathbf{T}} + \boldsymbol{\Psi})^{-1}\mathbf{D}$ and $\Sigma = \mathbf{I} - \mathbf{W}^{\mathbf{T}}(\mathbf{W}\mathbf{W}^{\mathbf{T}} + \boldsymbol{\Psi})^{-1}\mathbf{W}$, where $\mathbf{W} = \mathbf{B} \odot \mathbf{V}$.

Note that, in our sampling scheme, we considered the matrices $\mathbf{B}_x$ and $\mathbf{B}_y$ as simply parts of the big IBP matrix $\mathbf{B}$, and sampled them together using a single IBP draw. However, one could also sample them separately as two separate IBP matrices for $\mathbf{B}_x$ and $\mathbf{B}_y$. This would require different IBP draws for sampling $\mathbf{B}_x$ and $\mathbf{B}_y$ with some modification of the existing Gibbs sampler. Different IBP draws could result in different number of nonzero columns in $\mathbf{B}_x$ and $\mathbf{B}_y$. To deal with this issue, one could sample $\mathbf{B}_x$ (say having $K_x$ nonzero columns) and $\mathbf{B}_y$ (say having $K_y$ nonzero columns) first, introduce extra dummy columns ($|K_x - K_y|$ in number) in the matrix having smaller number of nonzero columns, and then set all such columns to zero. The effective $K$ for each iteration of the Gibbs sampler would be $\max\{K_x, K_y\}$. A similar scheme could also be followed for the corresponding real-valued matrices $\mathbf{V}_x$ and $\mathbf{V}_y$, sampling them in conjunction with $\mathbf{B}_x$ and $\mathbf{B}_y$ respectively.

## 4  Multitask Learning using Infinite CCA

Having set up the framework for infinite CCA, we now describe its applicability for the problem of multitask learning. In particular, we consider the setting when each example is associated with multiple labels. Here predicting each individual label becomes a task to be learned. Although one can individually learn a separate model for each task, doing this would ignore the label correlations. This makes borrowing the information across tasks crucial, making it imperative to share the statistical strength across all the task. With this motivation, we apply our infinite CCA model to capture the label correlations and to learn better predictive features from the data by projective it to a subspace directed by label information. It has been empirically and theoretically [25] shown that incorporating label information in dimensionality reduction indeed leads to better projections if the final goal is prediction.

More concretely, let $\mathbf{X} = [\mathbf{x}_1, \ldots, \mathbf{x}_N]$ be an $D \times N$ matrix of predictor variables, and $\mathbf{Y} = [\mathbf{y}_1, \ldots, \mathbf{y}_N]$ be an $M \times N$ matrix of the responses variables (i.e., the labels) with each $\mathbf{y}_i$ being an $M \times 1$ vector of responses for input $\mathbf{x}_i$. The labels can take real (for regression) or categorical (for classification) values. The infinite CCA model is applied on the pair $\mathbf{X}$ and $\mathbf{Y}$ which is akin to doing supervised dimensionality reduction for the inputs $\mathbf{X}$. Note that the generalized eigenvalue problem posed in such a supervised setting of CCA consists of cross-covariance matrix $\Sigma_{XY}$ and label covariance matrix $\Sigma_{YY}$. Therefore the projection takes into account both the input-output correlations and the label correlations. Such a subspace therefore is expected to consist of much better predictive features than one obtained by a naïve feature extraction approach such as simple PCA that completely ignores the label information, or approaches like Linear Discriminant Analysis (LDA) that do take into account label information but ignore label correlations.

Multitask learning using the infinite CCA model can be done in two settings: supervised and semi-supervised depending on whether or not the inputs of test data are involved in learning the shared subspace $\mathbf{Z}$.

## 4.1 Fully supervised setting

In the supervised setting, CCA is done on labeled data $(\mathbf{X}, \mathbf{Y})$ to give a single shared subspace $\mathbf{Z} \in \mathbb{R}^{K \times N}$ that is good across all tasks. A model is then learned in the $\mathbf{Z}$ subspace to learn $M$ task parameters $\{\theta_m\} \in \mathbb{R}^{K \times 1}$ where $m \in \{1, \dots, M\}$. Each of the parameters $\theta_m$ is then used to predict the labels for the test data of task $m$. However that since the test data is still $D$ dimensional, we need to either separately project it down onto the $K$ dimensional subspace and do predictions in this subspace, or "inflate" each task parameter back to $D$ dimensions by applying the projection matrix $\mathbf{W}_x$ and do predictions in the original $D$ dimensional space. The first option requires using the fact that $P(\mathbf{Z}|\mathbf{X}_{te}) \propto P(\mathbf{X}_{te}|\mathbf{Z})P(\mathbf{Z})$ which is a Gaussian $\mathcal{N}or(\mu_{Z|X}, \Sigma_{Z|X})$ with $\mu_{Z|X} = (\mathbf{W}_x^T \mathbf{\Psi}_x \mathbf{W}_x + \mathbf{I})^{-1} \mathbf{W}_x^T \mathbf{X}_{te}$ and $\Sigma_{Z|X} = (\mathbf{W}_x^T \mathbf{\Psi}_x \mathbf{W}_x + \mathbf{I})^{-1}$. With the second option, we can inflate each learned task parameter back to $D$ dimensions by applying the projection matrix $\mathbf{W}_x$. We choose the second option for the experiments. We call this fully supervised setting as model-1.

## 4.2 A Semi-supervised setting

In the semi-supervised setting, we combine training data and test data (with unknown labels) as $\mathbf{X} = [\mathbf{X}_{tr}, \mathbf{X}_{te}]$ and $\mathbf{Y} = [\mathbf{Y}_{tr}, \mathbf{Y}_{te}]$ where the labels $\mathbf{Y}_{te}$ are unknown. The infinite CCA model is then applied on the pair $(\mathbf{X}, \mathbf{Y})$ and the parts of $\mathbf{Y}$ consisting of $\mathbf{Y}_{te}$ are treated as a latent variables to be imputed. With this model, we get the embeddings also for the test data and thus training and testing both take place in the $K$ dimensional subspace, unlike model-1 in which training is done in $K$ dimensional subspace and prediction are made in the original $D$ dimensional subspace. We call this semi-supervised setting as model-2.

# 5 Experiments

Here we report our experimental results on several synthetic and real world datasets. We first show our results with the infinite CCA as a stand alone algorithm for CCA by using it on a synthetic dataset demonstrating its effectiveness in capturing the canonical correlations. We then also report our experiments on applying the infinite CCA model to the problem of multitask learning on two real world datasets.

## 5.1 Infinite CCA results on synthetic data

In the first experiment, we demonstrate the effectiveness of our proposed infinite CCA model in discovering the correct number of canonical correlation components, and in capturing the sparsity pattern underlying the projection matrix. For this, we generated two datasets of dimensions 25 and 10 respectively, with each having 100 samples. For this synthetic dataset, we knew the ground truth (i.e., the number of components, and the underlying sparsity of projection matrix). In particular, the dataset had 4 correlation components with a 63% sparsity in the true projection matrix. We then ran both classical CCA and infinite CCA algorithm on this dataset. Looking at *all* the correlations discovered by classical CCA, we found that it discovered 8 components having significant correlations, whereas our model correctly discovered exactly 4 components in the first place (we extract the MAP samples for $\mathbf{W}$ and $\mathbf{Z}$ output by our Gibbs sampler). Thus on this small dataset, standard CCA indeed seems to be finding spurious correlations, indicating a case of overfitting (the overfitting problem of classical CCA was also observed in [15] when comparing Bayesian versus classical CCA). Furthermore, as expected, the projection matrix inferred by the classical CCA had no exact zero entries and even after thresholding significantly small absolute values to zero, the uncovered sparsity was only about 25%. On the other hand, the projection matrix inferred by the infinite CCA model had 57% exact zero entries and 62% zero entries after thresholding very small values, thereby demonstrating its effectiveness in also capturing the sparsity patterns.

| Model | Yeast | | | | Scene | | | |
|---|---|---|---|---|---|---|---|---|
| | Acc | F1-macro | F1-micro | AUC | Acc | F1-macro | F1-micro | AUC |
| **Full** | 0.5583 | 0.3132 | 0.3929 | 0.5054 | 0.7565 | 0.3445 | 0.3527 | 0.6339 |
| **PCA** | 0.5612 | 0.3144 | 0.4648 | 0.5026 | 0.7233 | 0.2857 | 0.2734 | 0.6103 |
| **CCA** | 0.5441 | 0.2888 | 0.3923 | 0.5135 | 0.7496 | 0.3342 | 0.3406 | 0.6346 |
| **Model-1** | 0.5842 | 0.3327 | 0.4402 | 0.5232 | 0.7533 | 0.3630 | 0.3732 | 0.6517 |
| **Model-2** | **0.6156** | **0.3463** | **0.4954** | **0.5386** | **0.7664** | **0.3742** | **0.3825** | **0.6686** |

Table 1: Results on the multi-label classification task. Bold face indicates the best performance. Model-1 and Model-2 scores are averaged over 10 runs with different initializations.

## 5.2 Infinite CCA applied to multi-label prediction

In the second experiment, we use infinite CCA model to learn a set of related task in the context of multi-label prediction. For our experiments, we use two real-world multi-label datasets (Yeast and Scene) from the UCI repository. The Yeast dataset consists of 1500 training and 917 test examples, each having 103 features. The number of labels (or tasks) per example is 14. The Scene dataset consists of 1211 training and 1196 test examples, each having 294 features. The number of labels per example for this dataset is 6. We compare the following models for our experiments.

- Full: Train separate classifiers (SVM) on the full feature set for each task.

- PCA: Apply PCA on training and test data and then train separate classifiers for each task in the low dimensional subspace. This baseline ignores the label information while learning the low dimensional subspace.

- CCA: Apply classical CCA on training data to extract the shared subspace, learn separate model (i.e., task parameters) for each task in this subspace, project the task parameters back to the original $D$ dimensional feature space by applying the projection $\mathbf{W}_x$, and do predictions on the test data in this feature pace.

- Model-1: Use our supervised infinite CCA model to learn the shared subspace using only the training data (see section 4.1).

- Model-2: Use our semi-supervised infinite CCA model to *simultaneously* learn the shared subspace for both training and test data (see section 4.2).

The performance metrics used are overall accuracy, F1-Macro, F1-Micro, and AUC (Area Under ROC Curve). For PCA and CCA, we chose $K$ that gives the best performance, whereas this parameter was learned automatically for both of our proposed models. The results are shown in Table-1. As we can see, both the proposed models do better than the other baselines. Of the two proposed model, we see that model-2 does better in most cases suggesting that it is useful to incorporate the test data while learning the projections. This is possible in our probabilistic model since we could treat the unknown $\mathbf{Y}$'s of the test data as latent variables to be imputed while doing the Gibbs sampling.

We note here that our results are with cases where we only had access to small number of related task (yeast has 14, scene has 6). We expect the performance improvements to be even more significant when the number of (related) tasks is high.

## 6 Related Work

A number of approaches have been proposed in the recent past for the problem of supervised dimensionality reduction of *multi-label* data. The few approaches that exist include Partial Least Squares [2], multi-label informed latent semantic indexing [24], and multi-label dimensionality reduction using dependence maximization (MDDM) [26]. None of these, however, deal with the case when the data is only partially labeled. Somewhat similar in spirit to our approach is the work on supervised probabilistic PCA [25] that extends probabilistic PCA to the setting when we also have access to labels. However, it assumes a fixed number of components and does not take into account sparsity of the projections.

The CCA based approach to supervised dimensionality reduction is more closely related to the notion of dimension reduction for regression (DRR) which is formally defined as finding a low dimensional representation $\mathbf{z} \in \mathbb{R}^K$ of inputs $\mathbf{x} \in \mathbb{R}^D$ ($K \ll D$) for predicting multivariate outputs $\mathbf{y} \in \mathbb{R}^M$. An important notion in DRR is that of sufficient dimensionality reduction (SDR) [10, 8] which states that given $\mathbf{z}$, $\mathbf{x}$ and $\mathbf{y}$ are conditionally independent, i.e., $\mathbf{x} \perp\!\!\!\perp \mathbf{y}|\mathbf{z}$. As we can see in the graphical model shown in figure-1, the probabilistic interpretation of CCA yields the same condition with $\mathbf{X}$ and $\mathbf{Y}$ being conditionally independent given $\mathbf{Z}$.

Among the DRR based approaches to dimensionality reduction for real-valued multilabel data, Covariance Operator Inverse Regression (COIR) exploits the covariance structures of both the inputs and outputs [14]. Please see [14] for more details on the connection between COIR and CCA. Besides the DRR based approaches, the problem of extracting useful features from data, particularly with the goal of making predictions, has also been considered in other settings. The information bottleneck (IB) method [19] is one such example. Given input-output pairs $(\mathbf{X}, \mathbf{Y})$, the information bottleneck method aims to obtain a compressed representation $\mathbf{T}$ of $\mathbf{X}$ that can account for $\mathbf{Y}$. IB achieves this using a single tradeoff parameter to represent the tradeoff between the *complexity* of the representation of $\mathbf{X}$, measured by $I(\mathbf{X}; \mathbf{T})$, and the *accuracy* of this representation, measured by $I(\mathbf{T}; \mathbf{Y})$, where $I(.; .)$ denotes the mutual information between two variables. In another recent work [13], a joint learning framework is proposed which performs dimensionality reduction and multi-label classification simultaneously.

In the context of CCA as a stand-alone problem, sparsity is another important issue. In particular, sparsity improves model interpretation and has been gaining lots of attention recently. Existing works on sparsity in CCA include the double barrelled lasso which is based on a convex least squares approach [17], and CCA as a sparse solution to the generalized eigenvalue problem [18] which is based on constraining the cardinality of the solution to the generalized eigenvalue problem to obtain a sparse solution. Another recent solution is based on a direct greedy approach which bounds the correlation at each stage [22].

The probabilistic approaches to CCA include the works of [15] and [1], both of which use an automatic relevance determination (ARD) prior [5] to determine the number of relevant components, which is a rather ad-hoc way of doing this. In contrast, a nonparametric Bayesian alternative proposed here is a more principled to determine the number of components.

We note that the sparse factor analysis model proposed in [16] actually falls out as a special case of our proposed infinite CCA model if one of the datasets ($\mathbf{X}$ or $\mathbf{Y}$) is absent. Besides, the sparse factor analysis model is limited to factor analysis whereas the proposed model can be seen as an infinite generalization of both an unsupervised problem (sparse CCA), and (semi)supervised problem (dimensionality reduction using CCA with full or partial label information), with the latter being especially relevant for multitask learning in the presence of multiple labels.

Finally, multitask learning has been tackled using a variety of different approaches, primarily depending on what notion of task relatedness is assumed. Some of the examples include tasks generated from an IID space [4], and learning multiple tasks using a hierarchical prior over the task space [23, 7], among others. In this work, we consider multi-label prediction in particular, based on the premise that that a set of such related tasks share an underlying low-dimensional feature space [12] that captures the task relatedness.

## 7 Conclusion

We have presented a nonparametric Bayesian model for the Canonical Correlation Analysis problem to discover the dependencies between a set of variables. In particular, our model does not assume a fixed number of correlation components and this number is determined automatically based only on the data. In addition, our model enjoys sparsity making the model more interpretable. The probabilistic nature of our model also allows dealing with missing data. Finally, we also demonstrate the model's applicability to the problem of multi-label learning where our model, directed by label information, can be used to automatically extract useful predictive features from the data.

**Acknowledgements**

We thank the anonymous reviewers for helpful comments. This work was partially supported by NSF grant IIS-0712764.

# References

[1] C. Archambeau and F. Bach. Sparse probabilistic projections. In *Neural Information Processing Systems 21*, 2008.

[2] J. Arenas-García, K. B. Petersen, and L. K. Hansen. Sparse kernel orthonormalized pls for feature extraction in large data sets. In *Neural Information Processing Systems 19*, 2006.

[3] F. R. Bach and M. I. Jordan. A Probabilistic Interpretation of Canonical Correlation Analysis. In *Technical Report 688, Dept. of Statistics*. University of California, 2005.

[4] J. Baxter. A Model of Inductive Bias Learning. *Journal of Artificial Intelligence Research*, 12:149–198, 2000.

[5] C. M. Bishop. Bayesian PCA. In *Neural Information Processing Systems 11*, Cambridge, MA, USA, 1999. MIT Press.

[6] R. Caruana. Multitask Learning. *Machine Learning*, 28(1):41–75, 1997.

[7] H. Daumé III. Bayesian Multitask Learning with Latent Hierarchies. In *Conference on Uncertainty in Artificial Intelligence*, Montreal, Canada, 2009.

[8] K. Fukumizu, F. R. Bach, and M. I. Jordan. Dimensionality reduction for supervised learning with reproducing kernel hilbert spaces. *J. Mach. Learn. Res.*, 5:73–99, 2004.

[9] Z. Ghahramani, T. L. Griffiths, and P. Sollich. Bayesian Nonparametric Latent Feature Models. In *Bayesian Statistics 8. Oxford University Press*, 2007.

[10] A. Globerson and N. Tishby. Sufficient dimensionality reduction. *J. Mach. Learn. Res.*, 3:1307–1331, 2003.

[11] H. Hotelling. Relations Between Two Sets of Variables. *Biometrika*, pages 321–377, 1936.

[12] S. Ji, L. Tang, S. Yu, and J. Ye. Extracting Shared Subspace for Multi-label Classification. 2008.

[13] S. Ji and J. Ye. Linear dimensionality reduction for multi-label classification. In *Twenty-first International Joint Conference on Artificial Intelligence*, 2009.

[14] M. Kim and V. Pavlovic. Covariance operator based dimensionality reduction with extension to semi-supervised settings. In *Twelfth International Conference on Artificial Intelligence and Statistics*, Florida USA, 2009.

[15] A. Klami and S. Kaski. Local dependent components. In *ICML '07: Proceedings of the 24th international conference on Machine learning*, 2007.

[16] P. Rai and H. Daumé III. The infinite hierarchical factor regression model. In *Neural Information Processing Systems 21*, 2008.

[17] D. Hardoon J. Shawe-Taylor. The Double-Barrelled LASSO (Sparse Canonical Correlation Analysis). In *Workshop on Learning from Multiple Sources (NIPS)*, 2008.

[18] B. Sriperumbudur, D. Torres, and G. Lanckriet. The Sparse Eigenvalue Problem. In *arXiv:0901.1504v1*, 2009.

[19] N. Tishby, F. C. Pereira, and W. Bialek. The information bottleneck method. In *Proc. of the 37-th Annual Allerton Conference on Communication, Control and Computing*, pages 368–377.

[20] N. Ueda and K. Saito. Parametric Mixture Models for Multi-labeled Text. *Advances in Neural Information Processing Systems*, pages 737–744, 2003.

[21] C. Wang. Variational Bayesian approach to Canonical Correlation Analysis. In *IEEE Transactions on Neural Networks*, 2007.

[22] A. Wiesel, M. Kliger, and A. Hero. A Greedy Approach to Sparse Canonical Correlation Analysis. In *arXiv:0801.2748*, 2008.

[23] Y. Xue, X. Liao, L. Carin, and B. Krishnapuram. Multi-task Learning for Classification with Dirichlet Process Priors. *The Journal of Machine Learning Research*, 8:35–63, 2007.

[24] K. Yu, S. Yu, and V. Tresp. Multi-label Informed Latent Semantic Indexing. In *Proceedings of the 28th annual international ACM SIGIR conference on Research and development in information retrieval*, pages 258–265. ACM New York, NY, USA, 2005.

[25] S. Yu, K. Yu, V. Tresp, H. Kriegel, and M. Wu. Supervised Probabilistic Principal Component Analysis. In *KDD '06: Proceedings of the 12th ACM SIGKDD international conference on Knowledge discovery and data mining*, 2006.

[26] Y. Zhang Z. H. Zhou. Multi-Label Dimensionality Reduction via Dependence Maximization. In *Proceedings of the Twenty-Third AAAI Conference on Artificial Intelligence, AAAI 2008*, pages 1503–1505, 2008.

